# Analog Circuits for Constrained Optimization

**John C. Platt** [1]
Computer Science Department, 256-80
California Institute of Technology
Pasadena, CA 91125

## ABSTRACT

This paper explores whether analog circuitry can adequately perform constrained optimization. Constrained optimization circuits are designed using the differential multiplier method. These circuits fulfill time-varying constraints correctly. Example circuits include a quadratic programming circuit and a constrained flip-flop.

## 1   INTRODUCTION

Converting perceptual and cognitive tasks into constrained optimization problems is a useful way of generating neural networks to solve those tasks. Researchers have used constrained optimization networks to solve the traveling salesman problem [Durbin, 1987] [Hopfield, 1985], to perform object recognition [Gindi, 1988], and to decode error-correcting codes [Platt, 1986].

Implementing constrained optimization in analog VLSI is advantageous, because an analog VLSI chip can solve a large number of differential equations in parallel [Mead, 1989]. However, analog circuits only approximate the desired differential equations. Therefore, we have built test circuits to determine whether analog circuits can fulfill user-specified constraints.

## 2   THE DIFFERENTIAL MULTIPLIER METHOD

The differential multiplier method (DMM) is a method for creating differential equations that perform constrained optimization. The DMM was originally proposed by [Arrow, 1958] as an economic model. It was used as a neural network by [Platt, 1987].

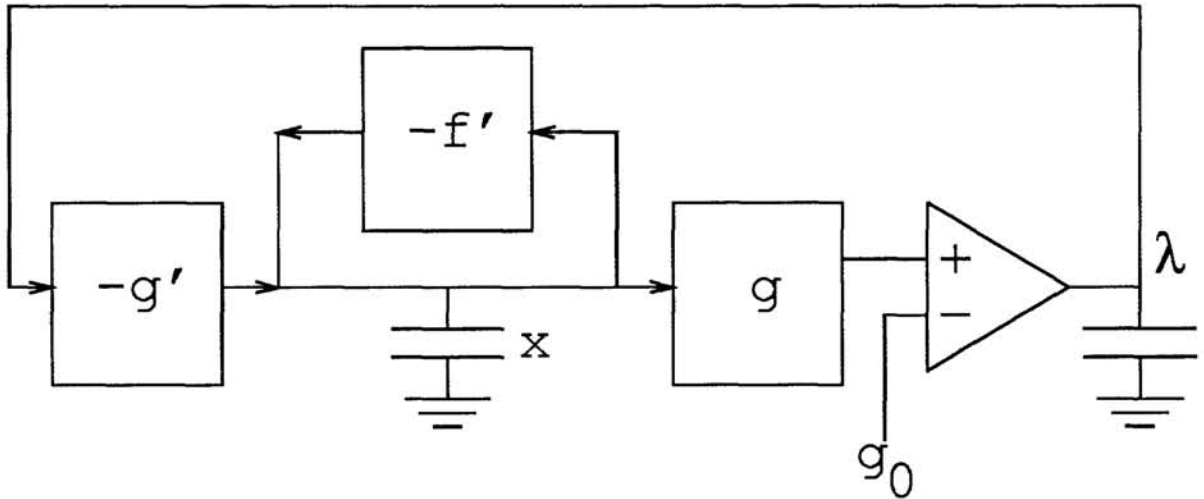

**Figure 1.** The architecture of the DMM. The $x$ capacitor in the figure represents the $x_i$ neurons in the network. The $-f'$ box computes the current needed for the neurons to minimize $f$. The rest of the circuitry causes the network to fulfill the constraint $g(\vec{x}) = 0$.

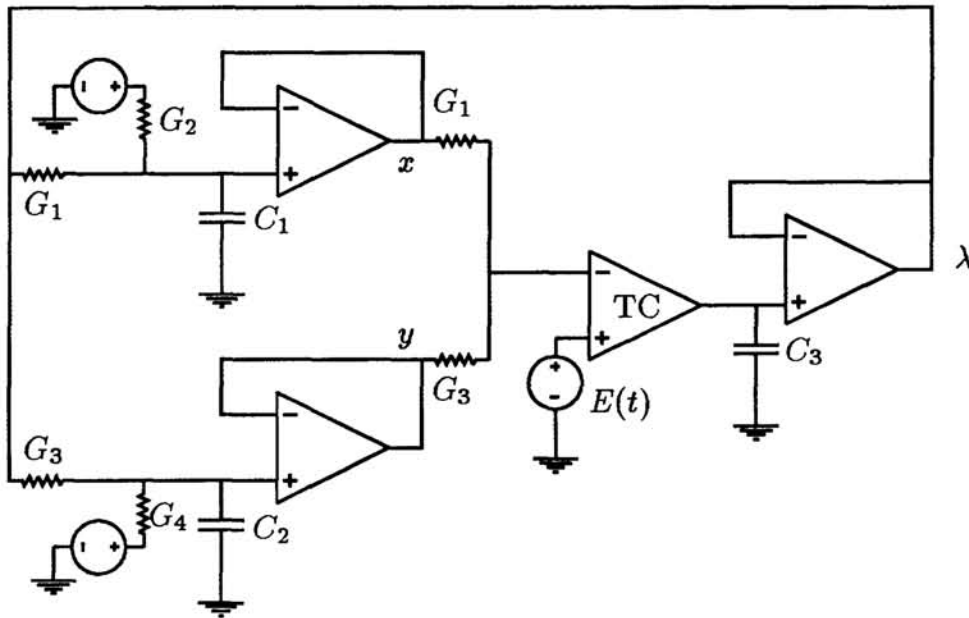

**Figure 2.** A circuit that implements quadratic programming. $x$, $y$, and $\lambda$ are voltages. "TC" refers to a transconductance amplifier.

A constrained optimization problem is find a $\vec{x}$ such that $f(\vec{x})$ is minimized subject to a constraint $g(\vec{x}) = 0$. In order to find a constrained minimum, the DMM finds the critical points $(\vec{x}, \lambda)$ of the Lagrangian

$$\mathcal{E} = f(\vec{x}) + \lambda g(\vec{x}), \tag{1}$$

by performing gradient *descent* on the variables $\vec{x}$ and gradient *ascent* on the Lagrange multiplier $\lambda$:

$$\begin{aligned}
\frac{dx_i}{dt} &= -\frac{\partial \mathcal{E}}{\partial x_i} = -\frac{\partial f}{\partial x_i} - \lambda \frac{\partial g}{\partial x_i}, \\
\frac{d\lambda}{dt} &= +\frac{\partial \mathcal{E}}{\partial \lambda} = g(\vec{x}).
\end{aligned} \tag{2}$$

The DMM can be thought of as a neural network which performs gradient descent on a function $f(\vec{x})$, plus feedback circuitry to find the $\lambda$ that causes the neural network output to fulfill the constraint $g(\vec{x}) = 0$ (see figure 1).

The gradient ascent on the $\lambda$ is necessary for stability. The stability can be examined by combining the two equations (2) to yield a set of second-order differential equations

$$\frac{d^2 x_i}{dt^2} + \sum_j \left( \frac{\partial^2 f}{\partial x_i \partial x_j} + \lambda \frac{\partial^2 g}{\partial x_i \partial x_j} \right) \frac{dx_j}{dt} + g \frac{\partial g}{\partial x_i} = 0, \tag{3}$$

which is analogous to the equations that govern a spring-mass-damping system. The differential equations (3) converge to the constrained minima if the damping matrix

$$M = \frac{\partial^2 f}{\partial x_i \partial x_j} + \lambda \frac{\partial^2 g}{\partial x_i \partial x_j} \tag{4}$$

is positive definite.

The DMM can be extended to satisfy multiple simultaneous constraints. The stability of the DMM can also be improved. See [Platt, 1987] for more details.

## 3    QUADRATIC PROGRAMMING CIRCUIT

This section describes a circuit that solves a specific quadratic programming problem for two variables. A quadratic programming circuit is interesting, because the basic differential multiplier method is guaranteed to find the constrained minimum. Also, quadratic programming is useful: it is frequently a sub-problem in a more complex task. A method of solving general nonlinear constrained optimization is sequential quadratic programming [Gill, 1981].

We build a circuit to solve a time-dependent quadratic programming problem for two variables:

$$\min A(x - x_0)^2 + B(y - y_0)^2, \tag{5}$$

subject to the constraint

$$Cx + Dy + E(t) = 0. \tag{6}$$

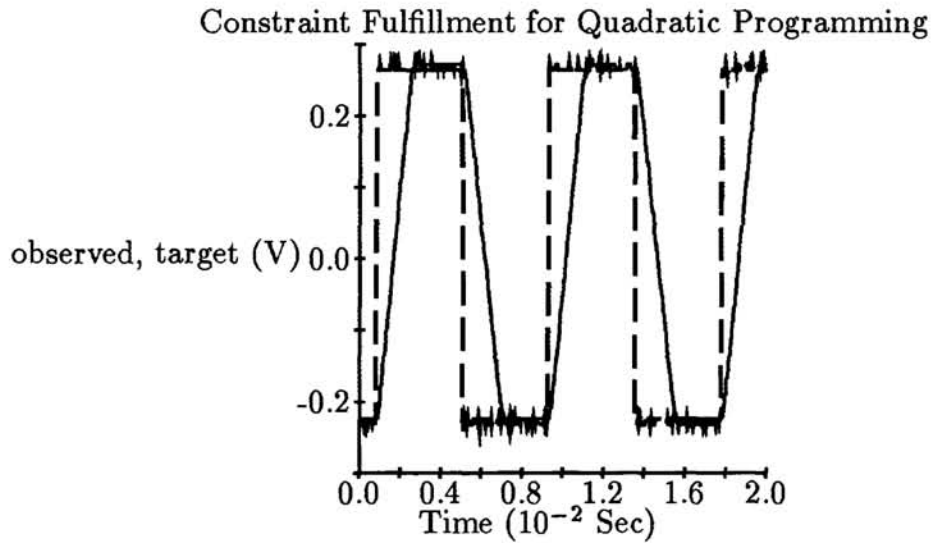

**Figure 3.** Plot of two input voltages of transconductance amplifier. The dashed line is the externally applied voltage $E(t)$. The solid line is the circuit's solution of $-Cx - Dy$. The constraint depends on time: the voltage $E(t)$ is a square wave. The linear constraint is fulfilled when the two voltages are the same. When $E(t)$ changes suddenly, the circuit changes $-Cx - Dy$ to compensate. The unusually shaped noise is caused by digitization by the oscilloscope.

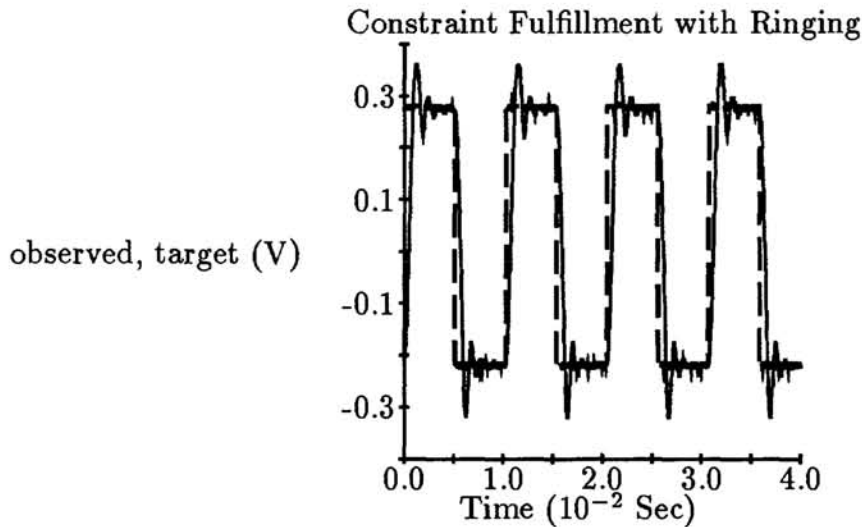

**Figure 4.** Plot of two input voltages of transconductance amplifier: the constraint forces are increased, which causes the system to undergo damped oscillations around the constraint manifold.

The basic differential multiplier method converts the quadratic programming problem into a system of differential equations:

$$k_1 \frac{dx}{dt} = -2Ax + 2Ax_0 - C\lambda,$$
$$k_2 \frac{dy}{dt} = -2By + 2By_0 - D\lambda, \qquad (7)$$
$$k_3 \frac{d\lambda}{dt} = Cx + Dy + E(t).$$

The first two equations are implemented with a resistor and capacitor (with a follower for zero output impedance). The third is implemented with resistor summing into the negative input of a transconductance amplifier. The positive input of the amplifier is connected to $E(t)$.

The circuit in figure 2 implements the system of differential equations

$$C_1 \frac{dx}{dt} = G_1(\lambda - x) + G_2(V_x - x),$$
$$C_2 \frac{dy}{dt} = G_3(\lambda - y) + G_4(V_y - y), \qquad (8)$$
$$C_3 \frac{d\lambda}{dt} = K \left( E(t) - \frac{G_1 x + G_3 y}{G_1 + G_3} \right),$$

where $K$ is the transconductance of the transconductance amplifier. The two systems of differential equations (7) and (8) can match with suitably chosen constants.

The circuit in figure 2 actually performs quadratic programming. The constraint is fulfilled when the voltages on the inputs of the transconductance amplifier are the same. The $g$ function is a difference between these voltages. Figure 3 is a plot of $-Cx - Dy$ and $E(t)$ as a function of time: they match reasonably well. The circuit in figure 2 therefore successfully fulfills the specified constraint.

Decreasing the capacitance $C_3$ changes the spring constant of the second-order differential equation. The forces that push the system towards the constraint manifold are increased without changing the damping. Therefore, the system becomes underdamped and the constraint is fulfilled with ringing (see figure 4).

The circuit in figure 2 can be easily expanded to solve general quadratic programming for $N$ variables: simply add more $x_i$ neurons, and interconnect them with resistors.

## 4  CONSTRAINED FLIP-FLOP

A flip-flop is two inverters hooked together in a ring. It is a bistable circuit: one inverter is on while the other inverter is off. A flip-flop can also be considered the simplest neural network: two neurons which inhibit each other.

If the inverters have infinite gain, then the flip-flop in figure 5 minimizes the function

$$\mathcal{E}_{\text{flip-flop}} = G_4 V_1 U_2 + G_2 V_2 U_1 - G_1 I_1 U_1 - G_3 I_2 U_2 + \frac{G_1 + G_2}{2} U_1^2 + \frac{G_3 + G_4}{2} U_2^2. \quad (9)$$

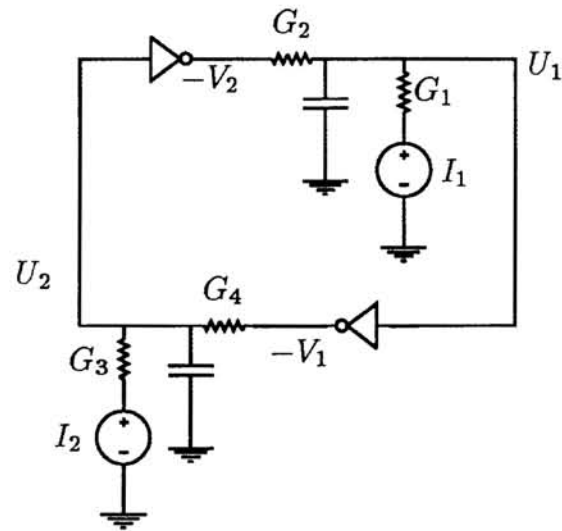

**Figure 5.** A flip-flop. $U_1$ and $U_2$ are voltages.

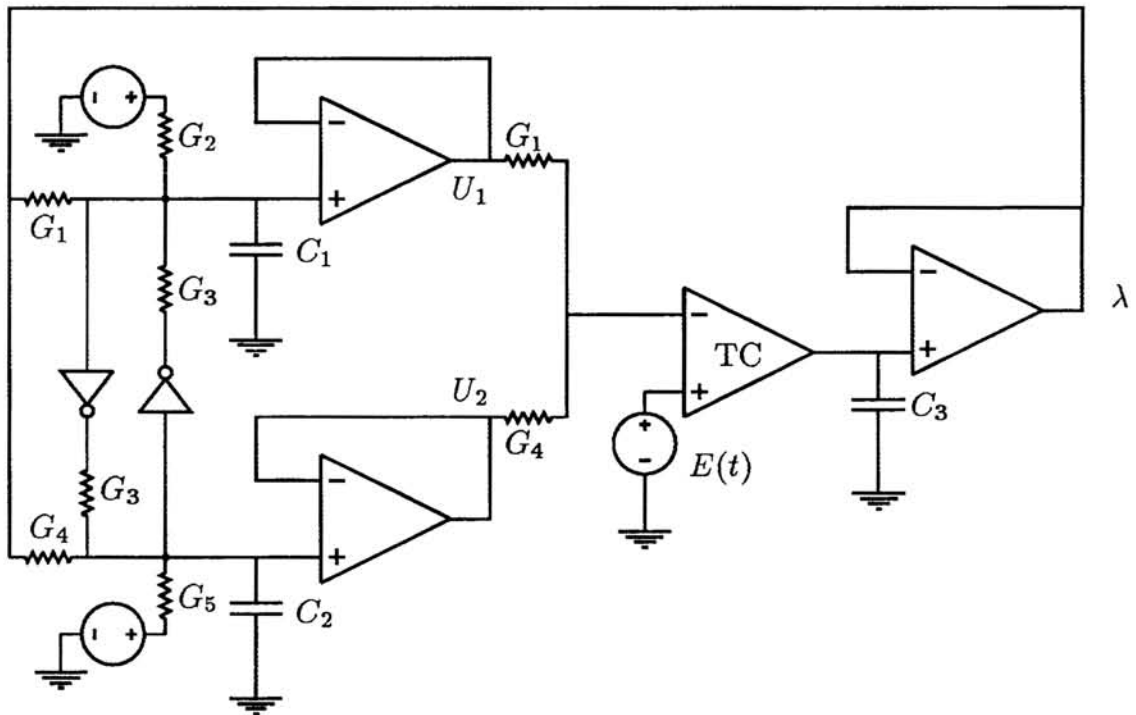

**Figure 6.** A circuit for constraining a flip-flop. $U_1$, $U_2$, and $\lambda$ are voltages.

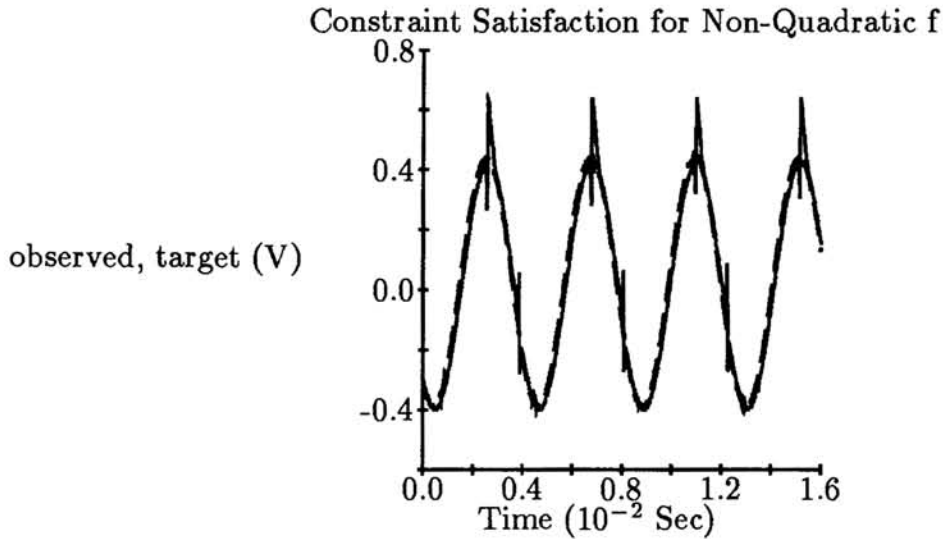

**Figure 7.** Constraint fulfillment for a non-quadratic optimization function. The plot consists of the two input voltages of the transconductance amplifier. Again, $E(t)$ is the dashed line and $-Cx - Dy$ is the solid line. The constraint is fulfilled when the two voltages are the same. As the constraint changes with time, the flip-flop changes state and the location of the constrained minimum changes abruptly. After the abrupt change, the constraint is temporarily not fulfilled. However, the circuit quickly fulfills the constraint. The temporary violation of the constraint causes the transient spikes in the $-Cx - Dy$ voltage.

Now, we can construct a circuit that minimizes the function in equation (9), subject to some linear constraint $Cx + Dy + E(t) = 0$, where $x$ and $y$ are the inputs to the inverters. The circuit diagram is shown in figure 6. Notice that this circuit is very similar to the quadratic programming circuit. Now, the $x$ and $y$ circuits are linked with a flip-flop, which adds non-quadratic terms to the optimization function.

The voltages $-Cx - Dy$ and $E(t)$ for this circuit are plotted in figure 7. For most of the time, $-Cx - Dy$ is close to the externally applied voltage $E(t)$. However, because $G_1 \neq G_4$ and $G_2 \neq G_5$, the flip-flop moves from one minima to the other and the constraint is temporarily violated. But, the circuitry gradually enforces the constraint again. The temporary constraint violation can be seen in figure 7.

## 5   CONCLUSIONS

This paper examines real circuits that have been constrained with the differential multiplier method. The differential multiplier method seems to work, even when the underlying circuit is non-linear, as in the case of the constrained flip-flop. Other papers examine applications of the differential multiplier method [Platt, 1987] [Gindi, 1988]. These applications could be built with the same parallel analog hardware discussed in this paper.

### Acknowledgement

This paper was made possible by funding from AT&T Bell Labs. Hardware was provided by Carver Mead, and Synaptics, Inc.

## Footnotes

[1]   Current address: Synaptics, 2860 Zanker Road, Suite 105, San Jose, CA 95134

### References

Arrow, K., Hurwicz, L., Uzawa, H., [1958], *Studies in Linear Nonlinear Programming*, Stanford University Press, Stanford, CA.

Durbin, R., Willshaw, D., [1987], "An Analogue Approach to the Travelling Salesman Problem," *Nature*, **326**, 689–691.

Gill, P. E., Murray, W., Wright, M. H., [1981], *Practical Optimization*, Academic Press, London.

Gindi, G, Mjolsness, E., Anandan, P., [1988], "Neural Networks for Model Matching and Perceptual Organization," *Advances in Neural Information Processing Systems I*, 618–625.

Hopfield, J. J., Tank, D. W., [1985], "'Neural' Computation of Decisions in Optimization Problems," *Biol. Cyber.*, **52**, 141–152.

Mead, C. A., [1989], *Analog VLSI and Neural Systems*, Addison-Wesley, Reading, MA.

Platt, J. C., Hopfield, J. J., [1986], "Analog Decoding with Neural Networks," *Neural Networks for Computing*, Snowbird, UT, 364–369.

Platt, J. C., Barr, A., [1987], "Constrained Differential Optimization," *Neural Information and Processing Systems*, 612–621.